# Slow, Decorrelated Features for Pretraining Complex Cell-like Networks

**James Bergstra**
University of Montreal
james.bergstra@umontreal.ca

**Yoshua Bengio**
University of Montreal
yoshua.bengio@umontreal.ca

## Abstract

We introduce a new type of neural network activation function based on recent physiological rate models for complex cells in visual area V1. A single-hidden-layer neural network of this kind of model achieves 1.50% error on MNIST. We also introduce an existing criterion for learning slow, decorrelated features as a pretraining strategy for image models. This pretraining strategy results in orientation-selective features, similar to the receptive fields of complex cells. With this pretraining, the same single-hidden-layer model achieves 1.34% error, even though the pretraining sample distribution is very different from the fine-tuning distribution. To implement this pretraining strategy, we derive a fast algorithm for online learning of decorrelated features such that each iteration of the algorithm runs in linear time with respect to the number of features.

## 1   Introduction

Visual area V1 is the first area of cortex devoted to handling visual input in the human visual system (Dayan & Abbott, 2001). One convenient simplification in the study of cell behaviour is to ignore the timing of individual spikes, and to look instead at their frequency. Some cells in V1 are described well by a linear filter that has been rectified to be non-negative and perhaps bounded. These so-called *simple cells* are similar to sigmoidal activation functions: their activity (firing frequency) is greater as an image stimulus looks more like some particular linear filter. However, these simple cells are a minority in visual area V1 and the characterization of the remaining cells there (and even beyond in visual areas V2, V4, MT, and so on) is a very active area of ongoing research. *Complex cells* are the next-simplest kind of cell. They are characterized by an ability to respond to narrow bars of light with particular orientations in some region (translation invariance) but to turn off when all those overlapping bars are presented at once. This non-linear response has been modeled by quadrature pairs (Adelson & Bergen, 1985; Dayan & Abbott, 2001): pairs of linear filters with the property that the sum of their squared responses is constant for an input image with particular spatial frequency and orientation (i.e. edges). It has also been modeled by max-pooling across two or more linear filters (Riesenhuber & Poggio, 1999). More recently, it has been argued that V1 cells exhibit a range of behaviour that blurs distinctions between simple and complex cells and between energy models and max-pooling models (Rust et al., 2005; Kouh & Poggio, 2008; Finn & Ferster, 2007).

Another theme in neural modeling is that cells do not react to single images, they react to image sequences. It is a gross approximation to suppose that each cell implements a function from image to activity level. Furthermore, the temporal sequence of images in a video sequence contains a lot of information about the invariances that we would like our models to learn. Throwing away that temporal structure makes learning about objects from images much more difficult. The principle of identifying slowly moving/changing factors in temporal/spatial data has been investigated by many (Becker & Hinton, 1993; Wiskott & Sejnowski, 2002; Hurri & Hyvärinen, 2003; Körding et al., 2004; Cadieu & Olshausen, 2009) as a principle for finding useful representations of images,

and as an explanation for why V1 simple and complex cells behave the way they do. A good overview can be found in (Berkes & Wiskott, 2005).

This work follows the pattern of initializing neural networks with unsupervised learning (*pretraining*) before *fine-tuning* with a supervised learning criterion. Supervised gradient descent explores the parameter space sufficiently to get low training error on smaller training sets (tens of thousands of examples, like MNIST). However, models that have been pretrained with appropriate unsupervised learning procedures (such as RBMs and various forms of auto-encoders) generalize better (Hinton et al., 2006; Larochelle et al., 2007; Lee et al., 2008; Ranzato et al., 2008; Vincent et al., 2008). See Bengio (2009) for a comprehensive review and Erhan et al. (2009) for a thorough experimental analysis of the improvements obtained. It appears that unsupervised pretraining guides the learning dynamics in better regions of parameter space associated with basins of attraction of the supervised gradient procedure corresponding to local minima with lower generalization error, even for very large training sets (unlike other regularizers whose effects tend to quickly vanish on large training sets) with millions of examples.

Recent work in the pretraining of neural networks has taken a generative modeling perspective. For example, the Restricted Boltzmann Machine is an undirected graphical model, and training it (by maximum likelihood) as such has been demonstrated to also be a good initialization. However, it is an interesting open question whether a better generative model is necessarily (or even typically) a better point of departure for fine-tuning. Contrastive divergence (CD) is not maximum likelihood, and works just fine as pretraining. Reconstruction error is an even poorer approximation of the maximum likelihood gradient, and sometimes works better than CD (with additional twists like sparsity or the denoising of (Vincent et al., 2008)).

The temporal coherence and decorrelation criterion is an alternative to training generative models such as RBMs or auto-encoder variants. Recently (Mobahi et al., 2009) demonstrated that a slowness criterion regularizing the top-most internal layer of a deep convolutional network during supervised learning helps their model to generalize better. Our model is similar in spirit to pre-training with the semi-supervised embedding criterion at each level (Weston et al., 2008; Mobahi et al., 2009), but differs in the use of decorrelation as a mechanism for preventing trivial solutions to a slowness criterion. Whereas RBMs and denoising autoencoders are defined for general input distributions, the temporal coherence and decorrelation criterion makes sense only in the context of data with slowly-changing temporal or spatial structure, such as images, video, and sound.

In the same way that simple cell models were the inspiration for sigmoidal activation units in artificial neural networks and validated simple cell models, we investigate in artificial neural network classifiers the value of complex cell models. This paper builds on these results by showing that the principle of temporal coherence is useful for finding initial conditions for the hidden layer of a neural network that biases it towards better generalization in object recognition. We introduce temporal coherence and decorrelation as a pretraining algorithm. Hidden units are initialized so that they are invariant to irrelevant transformations of the image, and sensitive to relevant ones. In order for this criterion to be useful in the context of large models, we derive a fast online algorithm for decorrelating units and maximizing temporal coherence.

## 2 Algorithm

### 2.1 Slow, decorrelated feature learning algorithm

(Körding et al., 2004) introduced a principle (and training criterion) to explain the formation of complex cell receptive fields. They based their analysis on the complex-cell model of (Adelson & Bergen, 1985), which describes a complex cell as a pair of half-rectified linear filters whose outputs are squared and added together and then a square root is applied to that sum.

Suppose $x$ is an input image and we have $F$ complex cells $h_1, ..., h_F$ such that $h_i = \sqrt{(u_i \cdot x)^2 + (v_i \cdot x)^2}$. (Körding et al., 2004) showed that by minimizing the following cost,

$$L_{K2004} = \alpha \sum_{i!=j} \frac{\text{Cov}_t(h_i, h_j)^2}{\text{Var}(h_i)\text{Var}(h_j)} + \sum_t \sum_i \frac{(h_{i,t} - h_{i,t-1})^2}{\text{Var}(h_i)} \tag{1}$$

over consecutive natural movie frames (with respect to model parameters), the filters $u_i$ and $v_i$ of each complex cell form local Gabor filters whose phases are offset by about 90 degrees, like the sine and cosine curves that implement a Fourier transform.

The criterion in Equation 1 requires a *batch* minimization algorithm because of the variance and covariance statistics that must be collected. This makes the criterion too slow for use with large datasets. At the same time, the size of the covariance matrix is quadratic in the number of features, so it is computationally expensive (perhaps prohibitively) to apply the criterion to train large numbers of features.

### 2.1.1   Online Stochastic Estimation of Covariance

This section presents an algorithm for approximately minimizing $L_{K2004}$ using an online algorithm whose iterations run in linear time with respect to the number of features. One way to apply the criterion to large or infinite datasets is by estimating the covariance (and variance) from consecutive minibatches of $N$ movie frames. Then the cost can be minimized by stochastic gradient descent.

We used an exponentially-decaying moving average to track the mean of each feature over time.

$$\bar{h}_i(t) = \rho \bar{h}_i(t-1) + (1-\rho)h_i(t)$$

For good results, $\rho$ should be chosen so that the estimates change very slowly. We used a value of $1.0 - 5.0 \times 10^{-5}$.

Then we estimated the variance of each feature over a minibatch like this:

$$\mathrm{Var}(h) \approx \frac{1}{N-1} \sum_{\tau=t}^{t+N-1} (h_i(t) - \bar{h}_i(t))^2$$

With this mean and variance, we computed normalized features for each minibatch:

$$z_i(t) = (h_i(t) - \bar{h}_i(t))/\sqrt{\mathrm{Var}(h) + 10^{-10}}$$

Letting $Z$ denote an $F \times N$ matrix with $N$ columns of $F$ normalized feature values, we estimate the correlation between features $h_i$ by the covariance in these normalized features: $C(t) = \frac{1}{N} Z(t)Z(t)'$.

We can now write down $L(t)$, a minibatch-wise approximation to Eq. 1:

$$L(t) = \alpha \sum_{i!=j} C_{ij}^2(t) + \sum_{\tau=0}^{N-1} \sum_i (z_i(t+\tau) - z_i(t+\tau-1))^2 \tag{2}$$

The time complexity of evaluating $L(t)$ from $Z$ using this expression is $O(FFN+NF)$. In practice we use small minibatches and our model has lots of features, so the fact that the time complexity of the algorithm is quadratic in $F$ is troublesome.

There is, however, a way to compute this value exactly in time linear in $F$. The key observation is that the sum of the squared elements of $C$ can be computed from the $N \times N$ *Gram* matrix $G(t) = Z(t)'Z(t)$.

$$\sum_{i=1}^{F} \sum_{j=1}^{F} C_{ij}^2(t) = \mathrm{Tr}(C(t)C(t))$$

$$= \frac{1}{N^2}\mathrm{Tr}(Z(t)Z(t)'Z(t)Z(t)') = \frac{1}{N^2}\mathrm{Tr}(Z(t)'Z(t)Z(t)'Z(t))$$

$$= \frac{1}{N^2}\mathrm{Tr}(G(t)G(t)) = \frac{1}{N^2}\mathrm{Tr}(G(t)G(t)')$$

$$= \frac{1}{N^2} \sum_{k=1}^{N} \sum_{l=1}^{N} G_{kl}^2(t) \doteq \frac{1}{N^2}|Z(t)'Z(t)|^2$$

Subtracting the $C_{ii}^2$ terms from the sum of all squared elements lets us rewrite Equation 2 in a way that suggests the linear-time implementation.

$$L(t) = \frac{\alpha}{N^2}\left(|Z(t)Z'(t)|^2 - \sum_{i=1}^{F}(\sum_{\tau=1}^{N} z_i(\tau)^2)^2\right) + \frac{1}{N-1}\sum_{\tau=1}^{N-1}\sum_{i=1}^{F}(z_i(\tau) - z_i(\tau-1))^2 \quad (3)$$

The time complexity of computing $L(t)$ using Equation 3 from $Z(t)$ is $O(NNF)$. The sum of squared correlations is still the most expensive term, but for the case where $N << F$, this expression makes $L(t)$'s computation linear in $F$. Considering that each iteration treats $N$ training examples, the per-training-example cost of this algorithm can be seen as $O(NF)$. In implementation, an additional factor of two in runtime can be obtained by only computing half of the Gram matrix $G$, which is symmetric.

## 2.2 Complex-cell activation function

Recently, (Rust et al., 2005) have argued that existing models, such as that of (Adelson & Bergen, 1985) cannot account for the variety of behaviour found in visual area V1. Some complex cells behave like simple cells to some extent and vice versa; there is a continuous range of simple to complex cells. They put forward a similar but more involved expression that can capture the simple and complex cells as special cases, but ultimately parameterizes a larger class of cell-response functions (Eq. 4).

$$a + \frac{\beta\left(\max(0, wx)^2 + \sum_{i=1}^{I}(u^{(i)}x)^2\right)^{\zeta} - \delta\left(\sum_{j=1}^{J}(v^{(j)}x)^2\right)^{\zeta}}{1 + \gamma\left(\max(0, wx)^2 + \sum_{i=1}^{I}(u^{(i)}x)^2\right)^{\zeta} + \epsilon\left(\sum_{j=1}^{J}(v^{(j)}x)^2\right)^{\zeta}} \quad (4)$$

The numerator in Eq 4 describes the difference between an *excitation* term and a *shunting inhibition* term. The denominator acts to normalize this difference. Parameters $w, u^{(i)}, v^{(j)}$ have the same shape as the input image $x$, and can be thought of as image filters like the first layer of a neural network or the codebook of a sparse-coding model. The parameters $a, \beta, \delta, \gamma, \epsilon, \zeta$ are scalars that control the range and shape of the activation function, given all the filter responses. The numbers $I$ and $J$ of quadratic filters required to explain a particular cellular response were on the order of 2-16.

We introduce the approximation in Equation 5 because it is easier to learn by gradient descent. We replaced the max operation with a softplus$(x) = \log(1 + e^x)$ function so that there is always a gradient on $w$ and $b$, even when $wx + b$ is negative. We fixed the scalar parameters to prevent the system from entering regimes of extreme non-linearity. We fixed $\beta, \delta, \gamma, \epsilon$ to 1, and $a$ to 0. We chose to fix the exponent $\zeta$ to 0.5 because (Rust et al., 2005) found that values close to 0.5 offered good fits to cell firing-rate data. Future work might look at choosing these constants in a principled way or adapting them; we found that these values worked well. The range of this activation function (as a function of $x$) is a connected set on the $(-1, 1)$ interval. However, the whole $(-1, 1)$ range is not always available, depending on the parameters. If the inhibition term is always 0 for example, then the activation function will be non-negative.

$$\frac{\sqrt{\log(1 + e^{wx+b})^2 + \sum_{i=1}^{I}(u^{(i)}x)^2} - \sqrt{\sum_{j=1}^{J}(v^{(j)}x)^2}}{1.0 + \sqrt{\log(1 + e^{wx+b})^2 + \sum_{i=1}^{I}(u^{(i)}x)^2} + \sqrt{\sum_{j=1}^{J}(v^{(j)}x)^2}} \quad (5)$$

## 3 Results

Classification results were obtained by adding a logistic regression model on top of the features learned, and treating the resulting model as a single-hidden-layer neural network. The weights of the logistic regression were always initialized to zero.

All work was done on 28x28 images (MNIST-sized), using a model with 300 hidden units. Each hidden unit had one linear filter $w$, a bias $b$, two quadratic excitatory filters $u_1, u_2$ and two quadratic inhibitory filters $v_1, v_2$. The computational cost of evaluating each unit was thus five times the cost of evaluating a normal sigmoidal activation function of the form $\tanh(w'x + b)$.

### 3.1 Random initialization

As a baseline, our model parameters were initialized to small random weights and used as the hidden layer of a neural network. Training this randomly-initialized model by stochastic gradient descent yielded test-set performance of 1.56% on MNIST.

The filters learned by this procedure looked somewhat noisy for the most part, but had low-frequency trends. For example, some of the quadratic filters had small local Gabor-like filters. We believe that these phase-offset pairs of Gabor-like functions allow the units to implement some shift-invariant response to edges with a specific orientation (Fig. 1).

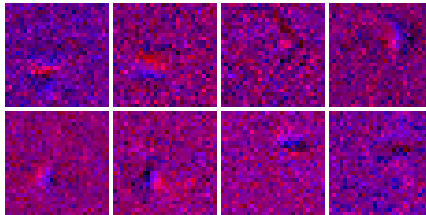

Figure 1: Four of the three hundred activation functions learned by training our model from random initialization to perform classification. Top row: the red and blue channels are the two quadratic filters of the excitation term. Bottom row: the red and blue channels are the two quadratic filters of the shunting inhibition term. Training approximately yields locally orientation-selective edge filters, opposite-orientation edges are inhibitory.

### 3.2 Pretraining with natural movies

Under the hypothesis that the matched Gabor functions (see Fig. 1) allowed our model to generalize better across slight translations of the image, we appealed to a pretraining process to initialize our model with values better than random noise.

We pretrained the hidden layer according to the online version of the cost in Eq. 3, using movies (MIXED-movies) made by sliding a 28 x 28 pixel window across large photographs. Each of these movies was short (just four frames long) and ten movies were used in each minibatch ($N = 40$). The sliding speed was sampled uniformly between 0.5 and 2 pixels per frame. The sliding direction was sampled uniformly from 0 to $2\pi$. The sliding initial position was sampled uniformly from image coordinates. Any sampled movie that slid off of the underlying image was rejected. We used two photographs to generate the movies. The first photograph was a grey-scale forest scene (resolution 1744x1308). The second photograph was a tiling of 100x100 MNIST digits (resolution 2800x2800). As a result of this procedure, digits are not at all centered in MIXED-movies: there might part of a '3' in the upper-left part of a frame, and part of a '7' in the lower right.

The shunting inhibition filters ($v_1, v_2$) learned after five hundred thousand movies (fifty thousand iterations of stochastic gradient descent) are shown in Figure 2. The filters learn to implement orientation-selective, shift-invariant filters at different spatial frequencies. The filters shown in figure 2 have fairly global receptive fields, but smaller more local receptive fields were obtained by applying $\ell_1$ weight-penalization during pretraining. The $\alpha$ parameter that balances decorrelation and slowness was chosen manually on the basis of the trained filters. We were looking for a diversity of filters with relatively low spatial frequency. The excitatory filters learned similar Gabor pairs but the receptive fields tended to be both smaller (more localized) and lower-frequency. Fine-tuning this pre-trained model with a learning rate of 0.003 with L1 weight decay of $10^{-5}$ yielded a test error rate of 1.34% on MNIST.

### 3.3 Pretraining with MNIST movies

We also tried pretraining with videos whose frames follow a similar distribution to the images used for fine-tuning and testing. We created MNIST movies by sampling an image from the training set, and moving around (translating it) according to a Brownian motion. The initial velocity was sampled from a zero-mean normal distribution with std-deviation 0.2. Changes in that velocity between each

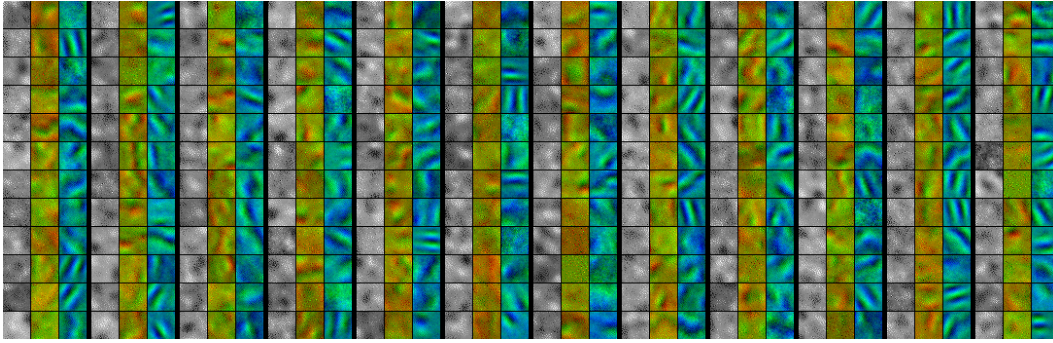

Figure 2: Filters from some of the units of the model, pretrained on small sliding image patches from two large images. The features learn to be direction-selective for moving edges by approximately implementing windowed Fourier transforms. These features have global receptive field, but become more local when an $\ell_1$ weight penalization is applied during pretraining. Excitatory filters looked similar, but tended to be more localized and with lower spatial frequency (fewer, shorter, broader stripes). Columns of the figure are arranged in triples: linear filter $w$ in grey, $u^{(1)}, u^{(2)}$ in red and green, $v^{(1)}, v^{(2)}$ in blue and green.

frame were sampled from zero-mean normal distribution with std-deviation 0.2. Furthermore, the digit image in each frame was modified according to a randomly chosen elastic deformation, as in (Loosli et al., 2007). As before, movies of four frames were created in this way and training was conducted on minibatches of ten movies ($N = 4 * 10 = 40$). Unlike the mnist frames in MIXED-movies, the frames of MNIST-movies contain a single digit that is approximately centered.

The activation functions learned by minimizing Equation 3 on these MNIST movies were qualitatively different from the activation functions learned from the MIXED movies. The inhibitory weights ($v_1, v_2$) learned from MNIST movies are shown in 3. Once again, the inhibitory weights exhibit the narrow red and green stripes that indicate edge-orientation selectivity. But this time they are not parallel straight stripes, they follow contours that are adapted to digit edges. The excitation filters $u_1, u_2$ were also qualitatively different. Instead of forming localized Gabor pairs, some formed large smooth blob-like shapes but most converged toward zero. Fine-tuning this pre-trained model with a learning rate of 0.003 with L1 weight decay of $10^{-5}$ yielded a test error rate of 1.37 % on MNIST.

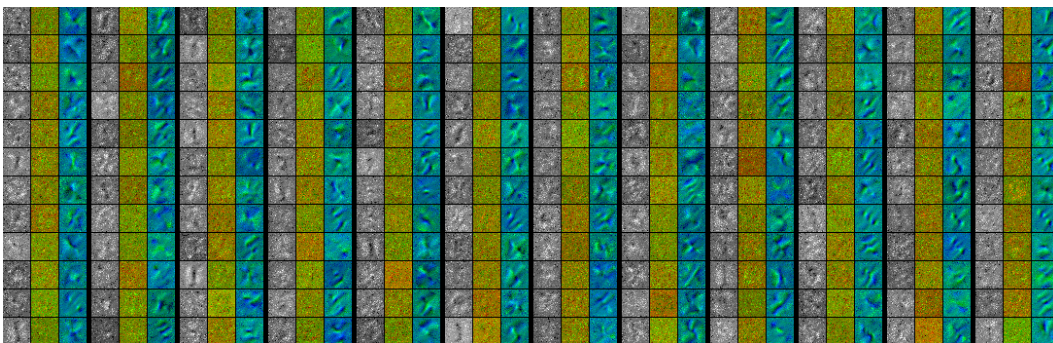

Figure 3: Filters of our model, pretrained on movies of centered MNIST training images subjected to Brownian translation. The features learn to be direction-selective for moving edges by approximately implementing windowed Fourier transforms. The filters are tuned to the higher spatial frequency in MNIST digits, as compared with the natural scene. Columns of the figure are arranged in triples: linear filter $w$ in grey, $u^{(1)}, u^{(2)}$ in red and green, $v^{(1)}, v^{(2)}$ in blue and green.

Table 1: Generalization error (% error) from 100 labeled MNIST examples after pretraining on MIXED-movies and MNIST-movies.

| Pre-training Dataset | Number of pretraining iterations ($\times 10^4$) | | | | | |
|---|---|---|---|---|---|---|
| | 0 | 1 | 2 | 3 | 4 | 5 |
| MIXED-movies | 23.1 | 21.2 | 20.8 | 20.8 | 20.6 | 20.6 |
| MNIST-movies | 23.1 | 19.0 | 18.7 | 18.8 | 18.4 | 18.6 |

## 4 Discussion

The results on MNIST compare well with many results in the literature. A single-hidden layer neural network of sigmoidal units can achieve 1.8% error by training from random initial conditions, and our model achieves 1.5% from random initial conditions. A single-hidden layer sigmoidal neural network pretrained as a denoising auto-encoder (and then fine-tuned) can achieve 1.4% error on average, and our model is able to achieve 1.34% error from many different fine-tuned models (Erhan et al., 2009). Gaussian SVMs trained just on the original MNIST data achieve 1.4%; our pretraining strategy allows our single-layer model be better than Gaussian SVMs (Decoste & Schölkopf, 2002). Deep learning algorithms based on denoising auto-encoders and RBMs are typically able to achieve slightly lower scores in the range of $1.2 - 1.3\%$ (Hinton et al., 2006; Erhan et al., 2009). The best convolutional architectures and models that have access to enriched datasets for fine-tuning can achieve classification accuriacies under $0.4\%$ (Ranzato et al., 2007). In future work, we will explore strategies for combining these methods and with our decorrelation criterion to train deep networks of models with quadratic input interactions. We will also look at comparative performance on a wider variety of tasks.

### 4.1 Transfer learning, the value of pretraining

To evaluate our unsupervised criterion of slow, decorrelated features as a pretraining step for classification by a neural network, we fine-tuned the weights obtained after ten, twenty, thirty, forty, and fifty thousand iterations of unsupervised learning. We used only a small subset (the first 100 training examples) from the MNIST data to magnify the importance of pre-training. The results are listed in Table 1. Training from random weights initial led to 23.1 % error. The value of pre-training is evident right away: after two unsupervised passes over the MNIST training data (100K movies and 10K iterations), the weights have been initialized better. Fine-tuning the weights learned on the MIXED-movies led to test error rate of 21.2%, and fine-tuning the weights learned on the MNIST-movies led to a test error rate of 19.0%. Further pretraining offers a diminishing marginal return, although after ten unsupervised passes through the training data (500K movies) there is no evidence of over-pretraining. The best score (20.6%) on MIXED-movies occurs at both eight and ten unsupervised passes, and the best score on MNIST-movies (18.4%) occurs after eight. A larger test set would be required to make a strong conclusion about a downward trend in test set scores for larger numbers of pretraining iterations. The results with MNIST-movies pretraining are slightly better than MIXED-movies but these results suggest strong transfer learning: the videos featuring digits in random locations and natural image patches are almost as good for pretraining as compared with videos featuring images very similar to those in the test set.

### 4.2 Slowness in normalized features encourages binary activations

Somewhat counter-intuitively, the slowness criterion requires movement in the features $h$. Suppose a feature $h_i$ has activation levels that are normally distributed around 0.1 and 0.2, but the activation at each frame of a movie is independent of previous frames. Since the features has a small variance, then the normalized feature $z_i$ will oscillate in the same way, but with unit variance. This will cause $z_i(t) - z_i(t-1)$ to be relatively high, and for our slowness criterion not to be well satisfied. In this way the lack of variance in $h_i$ can actually make for a relatively *fast* normalized feature $z_i$ rather than a slow one.

However, if $h_i$ has activation levels that are normally distributed around .1 and .2 for some image sequences and around .8 and .9 for other image sequences, the marginal variance in $h_i$ will be larger.

The larger marginal variance will make the oscillations between .1 and .2 lead to much smaller changes in the normalized feature $z_i(t)$. In this sense, the slowness objective can be maximally satisfied by features $h_i(t)$ that take near-minimum and near-maximum values for most movies, and never transition from a near-minimum to a near-maximum value during a movie.

When training on multiple short videos instead of one continuous one, it is possible for large changes in normalized-feature-activation never [or rarely] to occur during a video. Perhaps this is one of the roles of saccades in the visual system: to suspend the normal objective of temporal coherence during a rapid widespread change of activation levels.

### 4.3 Eigenvalue interpretation of decorrelation term

What does our unsupervised cost mean? One way of thinking about the decorrelation term (first term in Eq. 1) which helped us to design an efficient algorithm for computing it, is to think of it as flattening the eigen-spectrum of the correlation matrix of our features $h$ (over time). It is helpful to rewrite this cost in terms of normalized features: $z_i = \frac{h_i - \bar{h}_i}{\sigma_i}$, and to consider that we sum over all the elements of the correlation matrix including the diagonal.

$$\sum_{i \neq j} \frac{\text{Cov}_t(h_i, h_j)^2}{\text{Var}(h_i)\text{Var}(h_j)} = 2 \sum_{i=1}^{F-1} \sum_{j=i+1}^{F} \text{Cov}_t(z_i, z_j)^2 = \left( \sum_{i=1}^{F} \sum_{j=1}^{F} \text{Cov}_t(z_i, z_j)^2 \right) - F$$

If we use $C$ to denote the matrix whose $i, j$ entry is $\text{Cov}_t(z_i, z_j)$, and we use $U'\Lambda U$ to denote the eigen-decomposition of $C$, then we can transform this sum over $i \neq j$ further.

$$(\sum_{i=1}^{F} \sum_{j=1}^{F} \text{Cov}_t(z_i, z_j)^2) - F = \text{Tr}(C'C) - F = \text{Tr}(CC) - F$$

$$= \text{Tr}(U'\Lambda UU'\Lambda U) - F = \text{Tr}(UU'\Lambda UU'\Lambda) - F = \sum_{k=1}^{F} \Lambda_k^2 - F$$

We can interpret the first term of Eq. 1 as penalizing the squared eigenvalues of the covariance matrix between features in a normalized feature space ($z$ as opposed to $h$), or as minimizing the squared eigenvalues of the correlation matrix between features $h$.

## 5 Conclusion

We have presented an activation function for use in neural networks that is a simplification of a recent rate model of visual area V1 complex cells. This model learns shift-invariant, orientation-selective edge filters from purely supervised training on MNIST and achieves lower generalization error than conventional neural nets.

Temporal coherence and decorrelation has been put forward as a principle for explaining the functional behaviour of visual area V1 complex cells. We have described an online algorithm for minimizing correlation that has linear time complexity in the number of hidden units. Pretraining our model with this unsupervised criterion yields even lower generalization error: better than Gaussian SVMs, and competitive with deep denoising auto-encoders and 3-layer deep belief networks. The good performance of our model compared with poorer approximations of V1 is encouraging machine learning research inspired by neural information processing in the brain. It also helps to validate the corresponding computational neuroscience theories by showing that these neuron activations and unsupervised criteria have value in terms of learning.

### Acknowledgments

This research was performed thanks to funding from NSERC, MITACS, and the Canada Research Chairs.

# References

Adelson, E. H., & Bergen, J. R. (1985). Spatiotemporal energy models for the perception of motion. *Journal of the Optical Society of America*, *2*, 284–99.

Becker, S., & Hinton, G. E. (1993). Learning mixture models of spatial coherence. *Neural Computation*, *5*, 267–277.

Bengio, Y. (2009). Learning deep architectures for AI. *Foundations and Trends in Machine Learning*, *to appear*.

Berkes, P., & Wiskott, L. (2005). Slow feature analysis yields a rich repertoire of complex cell properties. *Journal of Vision*, *5*, 579–602.

Cadieu, C., & Olshausen, B. (2009). Learning transformational invariants from natural movies. In *Advances in neural information processing systems 21 (nips'08)*, 209–216. MIT Press.

Dayan, P., & Abbott, L. F. (2001). *Theoretical neuroscience*. The MIT Press.

Decoste, D., & Schölkopf, B. (2002). Training invariant support vector machines. *Machine Learning*, *46*, 161–190.

Erhan, D., Manzagol, P.-A., Bengio, Y., Bengio, S., & Vincent, P. (2009). The difficulty of training deep architectures and the effect of unsupervised pre-training. *AISTATS'2009* (pp. 153–160). Clearwater (Florida), USA.

Finn, I., & Ferster, D. (2007). Computational diversity in complex cells of cat primary visual cortex. *Journal of Neuroscience*, *27*, 9638–48.

Hinton, G. E., Osindero, S., & Teh, Y. (2006). A fast learning algorithm for deep belief nets. *Neural Computation*, *18*, 1527–1554.

Hurri, J., & Hyvärinen, A. (2003). Temporal coherence, natural image sequences, and the visual cortex. *Advances in Neural Information Processing Systems 15 (NIPS'02)* (pp. 141–148).

Körding, K. P., Kayser, C., Einhäuser, W., & König, P. (2004). How are complex cell properties adapted to the statistics of natural stimuli? *Journal of Neurophysiology*, *91*, 206–212.

Kouh, M. M., & Poggio, T. T. (2008). A canonical neural circuit for cortical nonlinear operations. *Neural Computation*, *20*, 1427–1451.

Larochelle, H., Erhan, D., Courville, A., Bergstra, J., & Bengio, Y. (2007). An empirical evaluation of deep architectures on problems with many factors of variation. *ICML 2007* (pp. 473–480). Corvallis, OR: ACM.

Lee, H., Ekanadham, C., & Ng, A. (2008). Sparse deep belief net model for visual area V2. In *Advances in neural information processing systems 20 (nips'07)*. Cambridge, MA: MIT Press.

Loosli, G., Canu, S., & Bottou, L. (2007). Training invariant support vector machines using selective sampling. In L. Bottou, O. Chapelle, D. DeCoste and J. Weston (Eds.), *Large scale kernel machines*, 301–320. Cambridge, MA.: MIT Press.

Mobahi, H., Collobert, R., & Weston, J. (2009). Deep learning from temporal coherence in video. *ICML 2009*. ACM. To appear.

Ranzato, M., Boureau, Y., & LeCun, Y. (2008). Sparse feature learning for deep belief networks. *NIPS 20*.

Ranzato, M., Poultney, C., Chopra, S., & LeCun, Y. (2007). Efficient learning of sparse representations with an energy-based model. *NIPS 19*.

Riesenhuber, M., & Poggio, T. (1999). Hierarchical models of object recognition in cortex. *Nature Neuroscience*, *2*, 1019–1025.

Rust, N., Schwartz, O., Movshon, J. A., & Simoncelli, E. (2005). Spatiotemporal elements of macaque V1 receptive fields. *Neuron*, *46*, 945–956.

Vincent, P., Larochelle, H., Bengio, Y., & Manzagol, P.-A. (2008). Extracting and composing robust features with denoising autoencoders. *ICML 2008* (pp. 1096–1103). ACM.

Weston, J., Ratle, F., & Collobert, R. (2008). Deep learning via semi-supervised embedding. *ICML 2008* (pp. 1168–1175). New York, NY, USA: ACM.

Wiskott, L., & Sejnowski, T. (2002). Slow feature analysis: Unsupervised learning of invariances. *Neural Computation*, *14*, 715–770.

